# Perceiving without Learning: from Spirals to Inside/Outside Relations

**Ke Chen\* and DeLiang L. Wang**
Department of Computer and Information Science and Center for Cognitive Science
The Ohio State University, Columbus, OH 43210-1277, USA
{kchen,dwang}@cis.ohio-state.edu

## Abstract

As a benchmark task, the spiral problem is well known in neural networks. Unlike previous work that emphasizes learning, we approach the problem from a generic perspective that does not involve learning. We point out that the spiral problem is intrinsically connected to the inside/outside problem. A generic solution to both problems is proposed based on oscillatory correlation using a time delay network. Our simulation results are qualitatively consistent with human performance, and we interpret human limitations in terms of synchrony and time delays, both biologically plausible. As a special case, our network without time delays can always distinguish these figures regardless of shape, position, size, and orientation.

## 1 INTRODUCTION

The *spiral* problem refers to distinguishing between a connected single spiral and disconnected double spirals, as illustrated in Fig. 1. Since Minsky and Papert (1969) first introduced the problem in their influential book on perceptrons, it has received much attention and has become a benchmark task in neural networks. Many solutions have been attempted using different learning models since Lang and Witbrock (1988) reported that the problem could not be solved with a standard multilayer perceptron. However, resulting learning systems are only able to produce decision regions highly constrained by the spirals defined in a training set, thus specific in shape, position, size, and orientation. Moreover, no explanation is provided as to why the problem is difficult for human subjects to solve. Grossberg and Wyse (1991) proposed a biologically plausible neural network architecture for figure-ground separation and reported their network can distinguish between connected and disconnected spirals. In their paper, however, no demonstration was given to the spiral problem, and their model does not exhibit the limitations that humans do.

There is a related problem in the study of visual perception, i.e., the perception of *inside/outside relations*. Considering the visual input of a single closed curve, the task of perceiving the inside/outside relation is to determine whether a specific pixel lies inside or outside the closed curve. For the human visual system, the perception of inside/outside relations often appears to be immediate and effortless (see an example in Fig. 2(a)). As illustrated in Fig. 2(b), however, the immediate perception is not available for humans when the bounding contour becomes highly convoluted (Ullman 1984). Ullman (1984) suggested the computation of spatial relation through the use of visual routines. Visual routines result in the conjecture that the inside/outside is inherently sequential. As pointed out recently by Ullman (1996), the processes underlying the perception of inside/outside relations are as yet unknown and applying visual routines is simply one alternative.

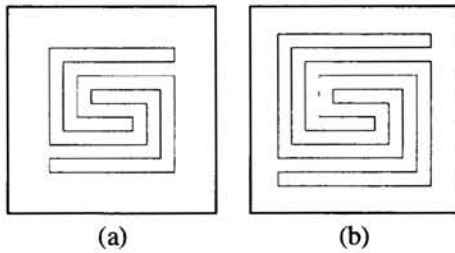

(a)          (b)

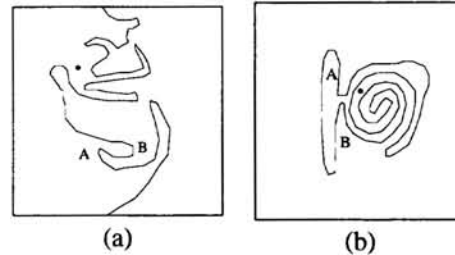

(a)          (b)

**Fig. 1**: The spiral problem. (a) a connected single spiral. (b) disconnected double spirals (adapted from Minsky and Papert 1969, 1988).

**Fig. 2**: Inside/Outside relations. (a) an example (adapted from Julesz 1995). (b) another example (adapted from Ullman 1984).

Theoretical investigations of brain functions indicate that timing of neuronal activity is a key to the construction of neuronal assemblies (Milner 1974, Malsburg 1981). In particular, the discovery of synchronous oscillations in the visual cortex (Singer & Gray 1995) has triggered much interest to develop computational models for oscillatory correlation. Recently, Terman and Wang (1995) proposed *locally excitatory globally inhibitory oscillator networks* (LEGION). They theoretically showed that LEGION can rapidly achieve both synchronization in a locally coupled oscillator group representing each object and desynchronization among a number of oscillator groups representing different objects. More recently, Campbell and Wang (1998) have studied time delays in networks of relaxation oscillators and analyzed the behavior of LEGION with time delays. Their studies show that loosely synchronous solutions can be achieved under a broad range of initial conditions and time delays. Therefore, LEGION provides a computational framework to study the process of visual perception from a standpoint of oscillatory correlation.

We explore both the spiral problem and the inside/outside relations by oscillatory correlation in this paper. We show that computation through LEGION with time delays yields a generic solution to these problems since time delays inevitably occur in information transmission of a biological system. This investigation indicates that perceptual performance would be limited if local activation cannot be rapidly propagated due to time delays. As a special case, LEGION without time delays reliably distinguishes between connected and disconnected spirals and discriminates the inside and the outside regardless of shape, position, size, and orientation. Thus, we suggest that this kind of problems may be better solved by a neural oscillator network rather than by sophisticated learning.

## 2 METHODOLOGY

The architecture of LEGION used in this paper is a two-dimensional network. Each oscillator is connected to its four nearest neighbors, and the global inhibitor (GI) receives excitation from each oscillator on the network and in turn inhibits each oscillator (Terman

& Wang 1995). In LEGION, a single oscillator, $i$, is defined as

$$\frac{dx_i}{dt} = 3x_i - x_i^3 - y_i + I_i + S_i + \rho \tag{1a}$$

$$\frac{dy_i}{dt} = \epsilon\Big(\lambda + \gamma \tanh(\beta x_i) - y_i\Big). \tag{1b}$$

Here $I_i$ represents external stimulation to the oscillator, and $S_i$ represents overall coupling from other oscillators and the GI in the network. The symbol $\rho$ denotes the amplitude of a Gaussian noise. Other parameters $\epsilon$, $\beta$, $\lambda$, and $\gamma$ are chosen to control a periodic solution of the dynamic system. The periodic solution alternates between the silent and the active phases of near steady-state behavior (Terman & Wang 1995). The coupling term $S_i$ at time $t$ is

$$S_i = \sum_{k \in N(i)} W_{ik} S_\infty(x_k(t - \tau), \theta_x) - W_z S_\infty(z, \theta_z), \tag{2}$$

where $S_\infty(x, \theta) = 1/(1 + \exp[-\kappa(x - \theta)])$ and the parameter $\kappa$ controls the steepness of the sigmoid function. $W_{ik}$ is a synaptic weight from oscillator $k$ to oscillator $i$, and $N(i)$ is the set of its immediate neighbors. $\tau$ is a time delay in interactions (Campbell & Wang 1998), and $\theta_x$ is a threshold over which an oscillator can affect its neighbors. $W_z$ is the positive weight used for the inhibition from the global inhibitor $z$, whose activity is defined as

$$\frac{dz}{dt} = \phi(\sigma_\infty - z). \tag{3}$$

where $\sigma_\infty = 0$ if $x_i < \theta_z$ for every oscillator $i$, and $\sigma_\infty = 1$ if $x_i(t) \geq \theta_z$ for at least one oscillator $i$. Here $\theta_z$ represents a threshold to determine whether the GI $z$ sends inhibition to oscillators, and the parameter $\phi$ determines the rate at which the inhibitor reacts to stimulation from oscillators.

We use *pattern formation* to refer to the behavior that all the oscillators representing the same object are synchronous, while the oscillators representing different objects are desynchronous. Terman and Wang (1995) have analytically shown that such a solution can be achieved in LEGION without time delays. However, a solution may not be achieved when time delays are introduced. Although the loose synchrony concept has been introduced to describe time delay behavior (Campbell & Wang 1998), it does not indicate pattern formation in an entire network even when loose synchrony is achieved because loose synchrony is a local concept defined in terms of pairs of neighboring oscillators. Here we introduce a measure called *min-max difference* in order to examine whether pattern formation is achieved. Suppose that oscillators $O_i$ and $O_j$ represent two pixels in the same object, and the oscillator $O_k$ represents a pixel in a different object. Moreover, let $t^s$ denote the time at which oscillator $O_s$ enters the active phase. The min-max difference measure is defined as $|t^i - t^j| < \tau_{RB}$ and $|t^i - t^k| \geq \tau_{RB}$, where $\tau_{RB}$ is the time period of an active phase. Intuitively, this measure suggests that pattern formation is achieved if any two oscillators representing two pixels in the same object have some overlap in the active phase, while any two oscillators representing two pixels belonging to different objects never stay in the active phase simultaneously. This definition of pattern formation applies to both exact synchrony in LEGION without time delays and loose synchrony with time delays.

## 3    SIMULATIONS

For a given image consisting of $N \times N$ pixels, a two-dimensional LEGION network with $N \times N$ oscillators is used so that each oscillator in the network corresponds to one pixel in the image. In the following simulations, the equations 1-3 were numerically solved using the fourth-order Runge-Kutta method. We illustrate stimulated oscillators with black squares. All oscillators were initialized randomly. A large number of simulations have

been conducted with a broad range of parameter values and network sizes (Chen & Wang 1997). Here we report typical results using a specific set of parameter values.

## 3.1  THE SPIRAL PROBLEM

For simulations, the two images in Fig. 1 were sampled as two binary images with $29 \times 29$ pixels. For these images, two problems can be addressed: (1) When an image is presented, can one determine whether it contains a single spiral or double spirals? (2) Given a point on a two-dimensional plane, can one determine whether it is inside or outside a specific spiral?

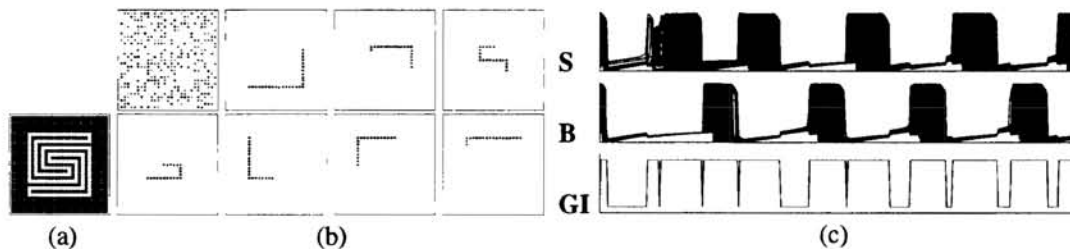

(a)                    (b)                              (c)

**Fig. 3**: Results of LEGION with a time delay $\tau = 0.002T$ ($T$ is the period of oscillation) for the spiral problem. The parameter values used in this simulation are $\epsilon = 0.003$, $\beta = 500$, $\gamma = 24.0$, $\lambda = 21.5$, $\alpha_T = 6.0$, $\rho = 0.03$, $\kappa = 500$, $\theta_x = -0.5$, $\theta_z = 0.1$, $\phi = 3.0$, $W_z = 1.5$, $I_s = 1.0$, and $I_u = -1.0$ where $I_s$ and $I_u$ are external input to stimulated and unstimulated oscillators, respectively.

We first applied LEGION with time delays to the single spiral image in Fig. 1(a). Fig. 3(a) illustrates the visual stimulus, where black pixels correspond to the stimulated oscillators and white ones correspond to the unstimulated oscillators. Fig. 3(b) shows a sequence of snapshots after the network was stabilized except for the first snapshot which shows the random initial state of the network. These snapshots are arranged in temporal order first from left to right and then from top to bottom. We observe from these snapshots that an activated oscillator in the spiral propagates its activation to its two immediate neighbors with some time delay, and the process of propagation forms a traveling wave along the spiral. We emphasize that, at any time, only the oscillators corresponding to a portion of the spiral stay in the active phase together, and the entire spiral can never be in the active phase simultaneously. Thus, based on the oscillatory correlation theory, our system cannot group the whole spiral together, which indicates that our system fails to realize that the pixels in the spiral belong to the same pattern. Note that the convoluted part of the background behaves similarly. Fig. 3(c) shows the temporal trajectories of the combined $x$ activities of the oscillators representing the spiral (S) and the background (B) as well as the temporal activity of the GI. According to the min-max difference measure, Fig. 3(c) shows that pattern formation cannot be achieved. In order to illustrate the effects of time delays, we applied LEGION without time delays to the same image. Simulation results show that pattern formation is achieved, and the single spiral can be segregated from the background by the second period (Chen & Wang 1997). Thus, LEGION without time delays can readily solve the spiral problem in this case. The failure to group the spiral in Fig. 3 is caused by time delays in the coupling of neighboring oscillators.

We also applied LEGION with time delays to the double spirals image in Fig. 1(b). Fig. 4(a) shows the visual stimulus. Fig. 4(b) shows a sequence of snapshots arranged in the same order as in Fig. 3(b). We observe from these snapshots that starting from an end of one spiral a traveling wave is formed along the spiral and the activated oscillators representing the spiral propagate their activation. Due to time delays, however, only the oscillators corresponding to a portion of the spiral stay in the active phase together, and the entire

spiral is never in the active phase simultaneously. The oscillators representing the other spiral have the same behavior. The results show that the pixels in any one of double spirals cannot be grouped as the same pattern. We mention that the behavior of our system for the convoluted part of the background is similar to that for the double spirals. It is also evident from Fig. 4(c) that the pattern formation is not achieved after the network was stabilized. We also applied LEGION without time delays to the double spirals image for the same purpose as described before. Simulation results also show that anyone of spirals can be segregated from both the other spiral and the background by the second period (Chen & Wang 1997). Once again, it indicates that the failure to group the double spirals in Fig. 4 results from time delays.

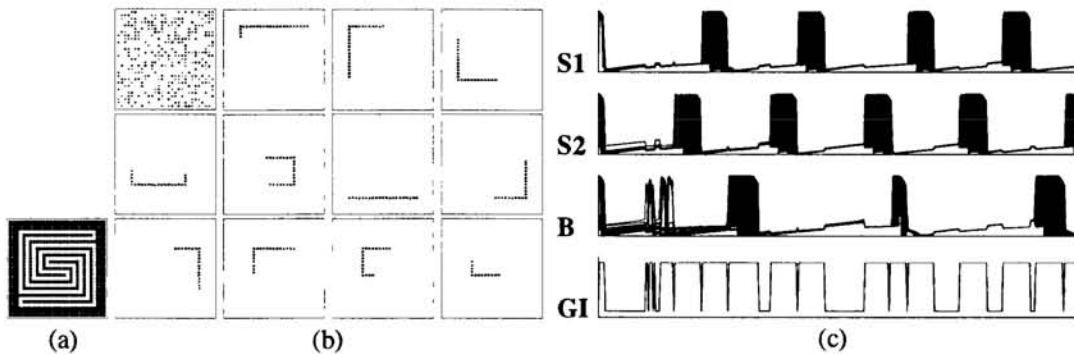

(a)                           (b)                                        (c)

**Fig. 4**: Results of LEGION without time delays for the spiral problem. The parameter values used are the same as listed in the caption of Fig. 3. In (c), **S1** and **S2** represent two disconnected spirals. **B** and **GI** denote background and the global inhibitor, respectively.

For the spiral problem, pattern formation means that solutions to the two problems in question can be provided to the questions of counting the number of objects or identifying whether two pixels belong to the same spiral or not. No such solutions are available when pattern formation is not achieved. Hence, our system cannot solve the spiral problem in general. Only under the special condition of no time delay can our system solve the problem.

## 3.2   INSIDE/OUTSIDE RELATIONS

For simulations, the two pictures in Fig. 2 were sampled as binary images with $43 \times 43$ pixels. We first applied LEGION with time delays to the two images in Fig. 2. Figures 5(a) and 6(a) show the visual stimuli, where black pixels represent areas A and B that correspond to stimulated oscillators and white pixels represent the boundary that corresponds to unstimulated oscillators. Figures 5(b) and 6(b) illustrate a sequence of snapshots after networks were stabilized except for the first snapshot which shows the random initial states of networks. Figures 5(c) and 6(c) show temporal trajectories of the combined $x$ activities of the oscillators representing areas A and B as well as the GI, respectively.

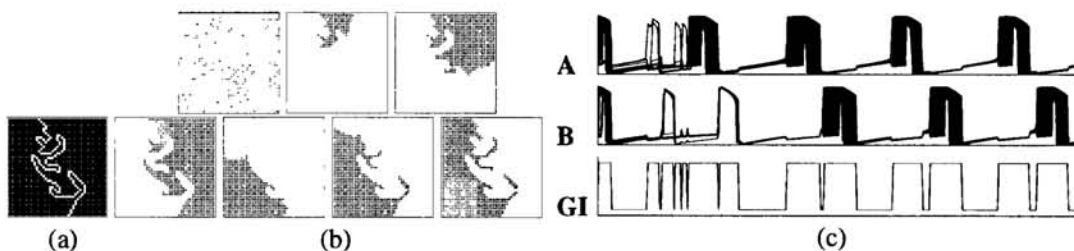

(a)                           (b)                                        (c)

**Fig. 5**: Results of LEGION with a time delay $\tau = 0.002T$ for Fig. 2(a). The parameter values used in this simulation are $\epsilon = 0.004$, $\gamma = 14.0$, $\lambda = 11.5$ and the other parameter values are the same

as listed in the caption of Fig. 3. In (c), **A**, **B**, and **GI** denote areas A, B, and the global inhibitor, respectively.

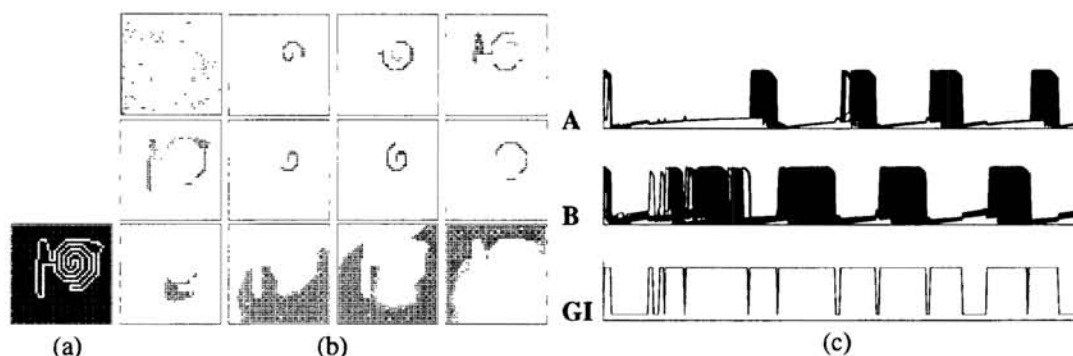

**Fig. 6**: Results of LEGION with a time delay $\tau = 0.002T$ for Fig. 2(b). The parameter values used and other statements are the same as listed in the caption of Fig. 5.

We observe from Fig. 5(b) that the activation of an oscillator can rapidly propagate through its neighbors to other oscillators representing the same area, and eventually all the oscillators representing the same area (A or B) stay together in the active phase simultaneously, though they generally enter the active phase at different times due to time delays. Thus, on the basis of oscillatory correlation, our system can group an entire area (A or B) together and recognize all the pixels in area A or B as elements of the same area. According to the min-max difference measure, Fig. 5(c) shows that pattern formation is achieved by the second period. In contrast, we observe from Fig. 6(b) that although an activated oscillator rapidly propagates its activation in open regions as shown in the last three snapshots, propagation is limited once the traveling wave spreads in spiral-like regions as shown in earlier snapshots. As a result, at any time, only the oscillators corresponding to a portion of either area stay in the active phase together, and the oscillators representing the whole area are never in the active phase simultaneously. Thus, on the basis of oscillatory correlation, our system cannot group the whole area, and fails to identify the pixels of one area as belonging to the same pattern. Furthermore, according to the min-max difference measure, Fig. 6(c) shows that pattern formation is not achieved after the network was stabilized. In order to illustrate the effects of time delays and show how to use an oscillator network to perceive inside/outside relations, we applied LEGION without time delays to the two images in Fig. 2. Our simulations show that LEGION without time delays readily segregates two areas in both cases by the second period (Chen & Wang 1997). Thus, the failure to group each area in Fig. 6 is also attributed to time delays in the coupling of neighboring oscillators. In general, the above simulations suggest that oscillatory correlation provides a way to address inside/outside relations by a neural network; when pattern formation is achieved, a single area segregates from other areas that appear in the same image. For a specific point on the two-dimensional plane, the inside/outside relations can be identified by examining whether the oscillator representing the point synchronizes with the oscillators representing a specific area or not.

## 4   DISCUSSION AND CONCLUSION

It has been reported that many neural network models can solve the spiral problem through learning. However, their solutions are subject to limitations because generalization abilities of resulting learning systems highly depend on the training set. As pointed out by Minsky and Papert (1969), solving the spiral problem is equivalent to detecting connectedness. They showed that connectedness cannot be computed by any diameter-limited or order-limited perceptrons (Minsky & Papert 1969). This limitation holds for multilayer perceptrons regardless of learning scheme (Minsky & Papert 1988, p.252). Unfortunately,

few people have discussed generality of their solutions. In contrast, our simulations have shown that LEGION without time delays can always distinguish these figures regardless of shape, position, size, and orientation. We emphasize that no learning is involved in LE-GION. In terms of performance, we suggest that the spiral problem may be better solved by a network of oscillators without learning.

Our system provides an alternative way to perceive inside/outside relations from a neural computation perspective. Our method is significantly distinguished from visual routines (Ullman 1984, 1996). First, the visual routine method is described as serial algorithms, while our system is an inherently parallel and distributed process although its emergent behavior reflects a degree of serial nature of the problems. Second, the visual routine method does not make a qualitative distinction between rapid effortless perception that corresponds to simple boundaries and slow effortful perception that corresponds to convoluted boundaries – the time a visual routine, e.g. the coloring method, takes varies continuously. In contrast, our system makes such a distinction: effortless perception with simple boundaries corresponds to when pattern formation is achieved, and effortful perception with convoluted boundaries corresponds to when pattern formation is not achieved. Third, perhaps more importantly conceptually, our system does not invoke high-level serial process to solve such problems like inside/outside relations; its solution involves the same mechanism as it does for parallel image segmentation (see Wang & Terman 1997).

**Acknowledgments**: Authors are grateful to S. Campbell for many discussions. This work was supported in part by an NSF grant (IRI-9423312), an ONR grant (N00014-93-10335), and an ONR Young Investigator Award (N00014-96-1-00676) to DLW.

## Footnotes

\*Also with National Laboratory of Machine Perception and Center for Information Science, Peking University, Beijing 100871, China. E-mail: chen@cis.pku.edu.cn

# References

Campbell, S. & Wang, D.L. (1998) Relaxation oscillators with time delay coupling. *Physica D* **111**:151-178.

Chen, K. & Wang, D.L. (1997) Perceiving without learning: from spirals to inside/outside relations. *Technical Report OSU-CISRC-8/97-TR38*, The Ohio State University.

Grossberg, S. & Wyse, L. (1991) A neural network architecture for figure-ground separation of connected scenic figures. *Neural Networks* **4**:723-742.

Julesz, B. (1995), *Dialogues on perception*. MIT Press.

Lang, K. & Witbrock, M. (1988) Learning to tell two spirals apart. *Proceeding of 1988 Connectionist Models Summer School*, pp. 52-59, Morgan Kaufmann.

Milner, P. (1974) A model for visual shape recognition. *Psychological Review* **81**:512-535.

Minsky, M. & Papert, R. (1969) *Perceptrons*. MIT Press.

Minsky, M. & Papert, R. (1988) *Perceptrons (extended version)*. MIT Press.

Singer, W. & Gray, C.M. (1995) Visual feature integration and the temporal correlation hypothesis. *Annual Review of Neuroscience* **18**:555-586.

Terman, D. & Wang, D.L. (1995) Global competition and local cooperation in a network of neural oscillators. *Physica D* **81**:148-176.

Ullman, S. (1984) Visual routines. *Cognition* **18**:97-159.

Ullman, S. (1996) *High-level vision*. MIT Press.

von der Malsburg, C. (1981) The correlation theory of brain function. *Internal Report 81-2*, Max-Planck-Institute for Biophysical Chemistry.

Wang, D.L. & Terman, D. (1997) Image segmentation based on oscillatory correlation. *Neural Computation* **9**:805-836.